# Learning Networks of Heterogeneous Influence

**Nan Du**[*]   **Le Song**[*]   **Alex Smola**[†]   **Ming Yuan**[*]
Georgia Institute of Technology[*],   Google Research[†]
dunan@gatech.edu  lsong@cc.gatech.edu
alex@smola.org  myuan@isye.gatech.edu

## Abstract

Information, disease, and influence diffuse over networks of entities in both natural systems and human society. Analyzing these transmission networks plays an important role in understanding the diffusion processes and predicting future events. However, the underlying transmission networks are often hidden and incomplete, and we observe only the time stamps when cascades of events happen. In this paper, we address the challenging problem of uncovering the hidden network only from the cascades. The structure discovery problem is complicated by the fact that the influence between networked entities is heterogeneous, which can not be described by a simple parametric model. Therefore, we propose a kernel-based method which can capture a diverse range of different types of influence without any prior assumption. In both synthetic and real cascade data, we show that our model can better recover the underlying diffusion network and drastically improve the estimation of the transmission functions among networked entities.

## 1   Introduction

Networks have been powerful abstractions for modeling a variety of natural and artificial systems that consist of a large collection of interacting entities. Due to the recent increasing availability of large-scale networks, network modeling and analysis have been extensively applied to study the spreading and diffusion of information, ideas, and even virus in social and information networks (see *e.g.*, [17, 5, 18, 1, 2]). However, the process of influence and diffusion often occurs in a hidden network that might not be easily observed and identified directly. For instance, when a disease spreads among people, epidemiologists can know only when a person gets sick, but they can hardly ever know where and from whom he (she) gets infected. Similarly, when consumers rush to buy some particular products, marketers can know when purchases occurred, but they cannot track in further where the recommendations originally came from [12]. In all such cases, we could observe only the time stamp when a piece of information has been received by a particular entity, but the exact path of diffusion is missing. Therefore, it is an interesting and challenging question whether we can uncover the diffusion paths based just on the time stamps of the events.

There are many recent studies on estimating correlation or causal structures from multivariate time-series data (see *e.g.*, [2, 6, 13]). However, in these models, time is treated as discrete index and not modeled as a random variable. In the diffusion network discovery problem, time is treated explicitly as a continuous variable, and one is interested in capturing how the occurrence of event at one node affects the time for its occurence at other nodes. This problem recently has been explored by a number of studies in the literature. Specifically, Meyers and Leskovec inferred the diffusion network by learning the infection probability between two nodes using a convex programming, called CONNIE [14]. Gomez-Rodriguez et al. inferred the network connectivity using a submodular optimization, called NETINF [4]. However, both CONNIE and NETINF assume that the transmission model for each pair of nodes is fixed with predefined transmission rate. Recently, Gomez-Rodriguez et al. proposed an elegant method, called NETRATE [3], using continuous temporal dynamics model to allow variable diffusion rates across network edges. NETRATE makes fewer number of assumptions and achieves better performance in various aspects than the previous two approaches. However, the limitation of NETRATE is that it requires the influence model on each edge to have a

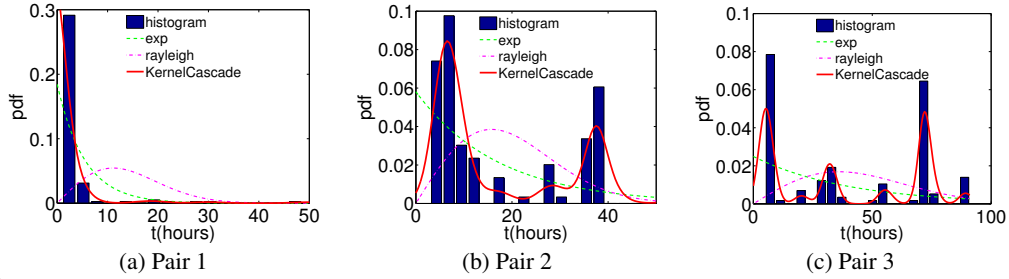

(a) Pair 1          (b) Pair 2          (c) Pair 3

Figure 1: The histograms of the interval between the time when a post appeared in one site and the time when a new post in another site links to it. Dotted and dash lines are density fitted by NETRATE. The solid lines are given by KernelCascade.

fixed parametric form, such as exponential, power-law, or Rayleigh distribution, although the model parameters learned from cascades could be different.

In practice, the patterns of information diffusion (or a spreading disease) among entities can be quite complicated and different from each other, going far beyond what a single family of parametric models can capture. For example, in twitter, an active user can be online for more than 12 hours a day, and he may instantly respond to any interesting message. However, an inactive user may just log in and respond once a day. As a result, the spreading pattern of the messages between the active user and his friends can be quite different from that of the inactive user.

Another example is from the information diffusion in a blogsphere: the hyperlinks between posts can be viewed as some kind of information flow from one media site to another, and the time difference between two linked posts reveal the pattern of diffusion. In Figure 1, we examined three pairs of media sites from the MemeTracker dataset [3, 9], and plotted the histograms of the intervals between the the moment when a post first appeared in one site and the moment when it was linked by a new post in another site. We can observe that information can have very different transmission patterns for these pairs. Parametric models fitted by NETRATE may capture the simple pattern in Figure 1(a), but they might miss the multimodal patterns in Figure 1(b) and Figure 1(c). In contrast, our method, called KernelCascade, is able to fit both data accurately and thus can handle the heterogeneity.

In the reminder of this paper, we present the details of our approach KernelCascade. Our key idea is to model the continuous information diffusion process using survival analysis by kernelizing the hazard function. We obtain a convex optimization problem with grouped lasso type of regularization and develop a fast block-coordinate descent algorithm for solving the problem. The sparsity patterns of the coefficients provide us the structure of the diffusion network. In both synthetic and real world data, our method can better recover the underlying diffusion networks and drastically improve the estimation of the transmission functions among networked entities.

## 2   Preliminary

In this section, we will present some basic concepts from survival analysis [7, 8], which are essential for our later modeling. Given a nonnegative random variable $T$ corresponding to the time when an event happens, let $f(t)$ be the probability density function of $T$ and $F(t) = Pr(T \leq t) = \int_0^t f(x)dx$ be its cumulative distribution function. The probability that an event does not happen up to time $t$ is thus given by the **survival function** $S(t) = Pr(T \geq t) = 1 - F(t)$. The survival function is a continuous and monotonically decreasing function with $S(0) = 1$ and $S(\infty) = \lim_{t \to \infty} S(t) = 0$. Given $f(t)$ and $S(t)$, we can define the instantaneous risk (or rate) that an event has not happened yet up to time $t$ but happens at time $t$ by the **hazard function**

$$h(t) = \lim_{\Delta t \to 0} \frac{Pr(t \leq T \leq t + \Delta t | T \geq t)}{\Delta t} = \frac{f(t)}{S(t)}. \tag{1}$$

With this definition, $h(t)\Delta t$ will be the approximate probability that an event happens in $[t, t + \Delta t)$ given that the event has not happened yet up to $t$. Furthermore, the hazard function $h(t)$ is also related to the survival function $S(t)$ via the differential equation $h(t) = -\frac{d}{dt} \log S(t)$, where we have used $f(t) = -S'(t)$. Solving the differential equation with boundary condition $S(0) = 1$, we can recover the survival function $S(t)$ and the density function $f(t)$ based on the hazard function $h(t)$, *i.e.*,

$$S(t) = \exp\left(-\int_0^t h(x)\,dx\right) \quad \text{and} \quad f(t) = h(t)\exp\left(-\int_0^t h(x)\,dx\right). \tag{2}$$

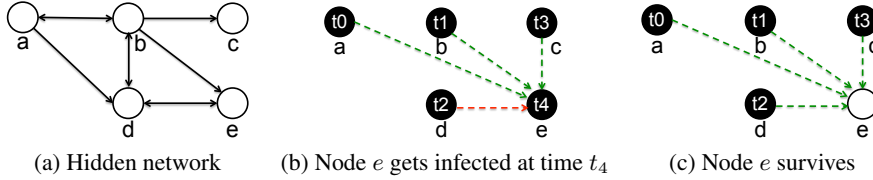

(a) Hidden network     (b) Node $e$ gets infected at time $t_4$     (c) Node $e$ survives

Figure 2: Cascades over a hidden network. Solid lines in panel(a) represent connections in a hidden network. In panel (b) and (c), filled circles indicate infected nodes while empty circles represent uninfected ones. Node $a$, $b$, $c$ and $d$ are the parents of node $e$ which got infected at $t_0 < t_1 < t_2 < t_3$ respectively and tended to infect node $e$. In panel (b), node $e$ survives given node $a$, $b$ and $c$ shown in green dash lines. However, it was infected by node $d$. In panel (c), node $e$ survives even though all its parents got infected.

## 3    Modeling Cascades using Survival Analysis

We use survival analysis to model information diffusion for networked entities. We will largely follow the presentation of Gomez-Rodriguez et al. [3], but add clarification when necessary. We assume that there is a fixed population of $N$ nodes connected in a directed network $\mathcal{G} = (\mathcal{V}, \mathcal{E})$. Neighboring nodes are allowed to directly influence each other. Nodes along a directed path may influence each other only through a diffusion process. Because the true underlying network is unknown, our observations are only the time stamps when events occur to each node in the network. The time stamps are then organized as cascades, each of which corresponds to a particular event. For instance, a piece of news posted on CNN website about "Facebook went public" can be treated as an event. It can spread across the blogsphere and trigger a sequence of posts from other sites referring to it. Each site will have a time stamp when this particular piece of news is being discussed and cited. The goal of the model is to capture the interplay between the hidden diffusion network and the cascades of observed event time stamps.

More formally, a directed edge, $j \to i$, is associated with an **transmission function** $f_{ji}(t_i|t_j)$, which is the conditional likelihood of an event happening to node $i$ at time $t_i$ given that the same event has already happened to node $j$ at time $t_j$. The transmission function attempts to capture the temporal dependency between the two successive events for node $i$ and $j$. In addition, we focus on shift-invariant transmission functions whose value only depends on the time difference, *i.e.*, $f_{ji}(t_i|t_j) = f_{ji}(t_i - t_j) = f_{ji}(\Delta_{ji})$ where $\Delta_{ji} := t_i - t_j$. Given the likelihood function, we can compute the corresponding survival function $S_{ji}(\Delta_{ji})$ and hazard function $h_{ji}(\Delta_{ji})$. When there is no directed edge $j \to i$, the transmission function and hazard function are both identically zeros, *i.e.*, $f_{ji}(\Delta_{ji}) = 0$ and $h_{ji}(\Delta_{ji}) = 0$, but the survival function is identically one, *i.e.*, $S_{ji}(\Delta_{ji}) = 1$. Therefore, the structure of the diffusion network is reflected in the non-zero patterns of a collection of transmission functions (or hazard functions).

A **cascade** is an $N$-dimensional vector $\boldsymbol{t}^c := (t_1^c, \ldots, t_N^c)^\top$ with $i$-th dimension recording the time stamp when event $c$ occurs to node $i$. Furthermore, $t_i^c \in [0, T^c] \cup \{\infty\}$, and the symbol $\infty$ labels nodes that have not been influenced during observation window $[0, T^c]$ — it does not imply that nodes are never influenced. The 'clock' is set to 0 at the start of each cascade. A dataset can contain a collection, $\mathcal{C}$, of cascades $\{\boldsymbol{t}^1, \ldots, \boldsymbol{t}^{|\mathcal{C}|}\}$. The time stamps assigned to nodes by a cascade induce a directed acyclic graph (DAG) by defining node $j$ as the parent of $i$ if $t_j < t_i$. Thus, it is meaningful to refer to *parents and children within a cascade* [3], which is different from the parent-child structural relation on the true underlying diffusion network. Since the true network is inferred from many cascades (each of which imposes its own DAG structure), the inferred network is typically not a DAG.

The likelihood $\ell(\boldsymbol{t}^c)$ of a cascade induced by event $c$ is then simply a product of all individual likelihood $\ell_i(\boldsymbol{t}^c)$ that event $c$ occurs to each node $i$. Depending on whether event $c$ actually occurs to node $i$ in the data, we can compute this individual likelihood as:

**Event $c$ did occur at node $i$.** We assume that once an event occurs at node $i$ under the influence of a particular parent $j$ in a cascade, the same event will not happen again. In Figure 2(b), node $e$ is susceptible given its parent $a$, $b$, $c$ and $d$. However, only node $d$ is the first parent who infects node $e$. Because each parent could be equally likely to first influence node $i$, the likelihood is just a simple *sum* over the likelihoods of the mutually disjoint events that node $i$ has survived from the influence of all the other parents except the first parent $j$, *i.e.*,

$$\ell_i^+(\boldsymbol{t}^c) = \sum_{j:t_j^c < t_i^c} f_{ji}(\Delta_{ji}^c) \prod_{k:k \neq j, t_k^c < t_i^c} S_{ki}(\Delta_{ki}^c) = \sum_{j:t_j^c < t_i^c} h_{ji}(\Delta_{ji}^c) \prod_{k:t_k^c < t_i^c} S_{ki}(\Delta_{ki}^c). \qquad (3)$$

**Event $c$ did not occur at node $i$.** In other words, node $i$ survives from the influence of all parents (see Figure 2(c) for illustration). The likelihood is a *product* of survival functions, *i.e.*,

$$\ell_i^-(\boldsymbol{t}^c) = \prod_{t_j \le T^c} S_{ji}(T^c - t_j). \tag{4}$$

Combining the above two scenarios together, we can obtain the overall likelihood of a cascade $\boldsymbol{t}^c$ by multiplying together all individual likelihoods, *i.e.*,

$$\ell(\boldsymbol{t}^c) = \underbrace{\prod_{t_i^c > T^c} \ell_i^-(\boldsymbol{t}^c)}_{\text{uninfected nodes}} \times \underbrace{\prod_{t_i^c \le T^c} \ell_i^+(\boldsymbol{t}^c)}_{\text{infected nodes}}. \tag{5}$$

Therefore, the likelihood of all cascades is a product of the these individual cascade likelihoods, *i.e.* $\ell(\{\boldsymbol{t}^1, \ldots, \boldsymbol{t}^{|\mathcal{C}|}\}) = \prod_{c=1,\ldots,|\mathcal{C}|} \ell(\boldsymbol{t}^c)$. In the end, we take the negative log of this likelihood function and regroup all terms associated with edges pointing to node $i$ together to derive

$$\mathcal{L}(\{\boldsymbol{t}^1, \ldots, \boldsymbol{t}^{|\mathcal{C}|}\}) = -\sum_i \left( \sum_j \sum_{\{c|t_j^c < t_i^c\}} \log S(\Delta_{ji}^c) + \sum_{\{c|t_i^c \leqslant T^c\}} \log \sum_{\{t_j^c < t_i^c\}} h(\Delta_{ji}^c) \right) \tag{6}$$

There are two interesting implications from this negative log likelihood function. First, the function can be expressed using only the hazard and the survival function. Second, the function is decomposed into additive contribution from each node $i$. We can therefore estimate the hazard and survival function for each node separately. Previously, Gomez-Rodriguez et al. [3] used parametric hazard and survival functions, and they estimated the model parameters using the convex programming. In contrast, we will instead formulate an algorithm using kernels and grouped parameter regularization, which allows us to estimate complicated hazard and survival functions without overfitting.

## 4    KernelCascade for Learning Diffusion Networks

This section presents our kernel method for uncovering diffusion networks from cascades. Our key idea is to kernelize the hazard function used in the negative log-likelihood in (6), and then estimate the parameters using grouped lasso type of optimization.

### 4.1    Kernelizing survival analysis

Kernel methods are powerful tools for generalizing classical linear learning approaches to analyze nonlinear relations. A kernel function, $k : \mathcal{X} \times \mathcal{X} \to \mathbb{R}$, is a real-valued positive definite symmetric function iff. for any set of points $\{\tau_1, \tau_2, \ldots, \tau_m\} \in \mathcal{X}$ the kernel matrix $K$ with entris $K_{ls} := k(\tau_l, \tau_s)$ is positive definite. We want to model heterogeneous transmission functions, $f_{ji}(\Delta_{ji})$, from $j$ to $i$. Rather than directly kernelizing the transmission function, we kernelize the hazard function instead, by assuming that it is a linear combination of $m$ kernel functions, *i.e.*,

$$h_{ji}(\Delta_{ji}) = \sum_{l=1}^m \alpha_{ji}^l k(\tau_l, \Delta_{ji}), \tag{7}$$

where we fix one argument of each kernel function, $k(\tau_l, \cdot)$, to a point $\tau_l$ in a uniform grid of $m$ locations in the range of $(0, \max_c T^c]$. To achieve fully nonparametric modeling of the hazard function, we can let $m$ grow as we see more cascades. Alternatively, we can also place a nonlinear basis function on each time point in the observed cascades. For efficiency consideration, we will use a fixed uniform grid in our later experiments. Since the hazard function is always positive, we use positive kernel functions and require the weights to be positive, *i.e.*, $k(\cdot, \cdot) \ge 0$ and $\alpha_{ji}^l \ge 0$ to capture such constraint. For simplicity of notation, we will define vectors $\boldsymbol{\alpha}_{ji} := (\alpha_{ji}^1, \ldots, \alpha_{ji}^m)^\top$, and $\boldsymbol{k}(\Delta_{ji}) := (k(\tau_1, \Delta_{ji}), \ldots, k(\tau_m, \Delta_{ji}))^\top$. Hence, the hazard function can be written as $h_{ji}(\Delta_{ji}) = \boldsymbol{\alpha}_{ji}^\top \boldsymbol{k}(\Delta_{ji})$.

In addition, the survival function and likelihood function can also be kernelized based on their respective relation with the hazard function in (2). More specifically, let $g_l(\Delta_{ji}) := \int_0^{\Delta_{ji}} k(\tau_l, x) dx$ and the corresponding vector $\boldsymbol{g}(\Delta_{ji}) := (g_1(\Delta_{ji}), \ldots, g_m(\Delta_{ji}))^\top$. We then can derive

$$S_{ji}(\Delta_{ji}) = \exp\left(-\boldsymbol{\alpha}_{ji}^\top \boldsymbol{g}(\Delta_{ji})\right) \quad \text{and} \quad f_{ji}(\Delta_{ji}) = \left(\boldsymbol{\alpha}_{ji}^\top \boldsymbol{k}(\Delta_{ji})\right) \exp\left(-\boldsymbol{\alpha}_{ji}^\top \boldsymbol{g}(\Delta_{ji})\right). \tag{8}$$

In the formulation, we need to perform integration over the kernel function to compute $g_l(\Delta_{ji})$. This can be done efficiently for many kernels, such as the Gaussian RBF kernel, the Laplacian kernel, the Quartic kernel, and the Triweight kernel. In later experiments, we mainly focus on the Gaussian

RBF kernel, $k(\tau_l, \tau_s) = \exp(-\|\tau_l - \tau_s\|^2/(2\sigma^2))$, and derive a closed form solution for $g_l(\Delta_{ji})$ as

$$g_l(\Delta_{ji}) = \int_0^{\Delta_{ji}} k(\tau_l, x)\, dx = \frac{\sqrt{2\pi}\sigma}{2}\left(\operatorname{erfc}\left(\frac{\tau_l - \Delta_{ji}}{\sqrt{2}\sigma}\right) - \operatorname{erfc}\left(\frac{\tau_l}{\sqrt{2}\sigma}\right)\right), \qquad (9)$$

where $\operatorname{erfc}(t) := \frac{2}{\sqrt{\pi}}\int_t^\infty e^{-x^2}\, dx$ is the error function. Yet, our method is not limited to the particular RBF kernel. If there is no closed form solution for the one-dimensional integration, we can use a large number of available numerical integration methods for this purpose [15]. We note that given a dataset, both the vector $\boldsymbol{k}(\Delta_{ji})$ and $\boldsymbol{g}(\Delta_{ji})$ need to be computed only once as a preprocessing, and then can be reused in the algorithm.

## 4.2 Estimating sparse diffusion networks

Next we plug in the kernelized hazard function and survival function into the likelihood of cascades in (6). Since the negative log likelihood is separable for each node $i$, we can optimize the set of variables $\{\boldsymbol{\alpha}_{ji}\}_{j=1}^N$ separately. As a result, the negative log likelihood for the data associated with node $i$ can be estimated as

$$\mathcal{L}_i\left(\{\boldsymbol{\alpha}_{ji}\}_{j=1}^N\right) = \sum_j \sum_{\{c|t_j^c < t_i^c\}} \boldsymbol{\alpha}_{ji}^\top \boldsymbol{g}(\Delta_{ji}^c) - \sum_{\{c|t_i^c \leqslant T^c\}} \log \sum_{\{t_j^c < t_i^c\}} \boldsymbol{\alpha}_{ji}^\top \boldsymbol{k}(\Delta_{ji}^c). \qquad (10)$$

A desirable feature of this function is that it is convex in its arguments, $\{\boldsymbol{\alpha}_{ji}\}_{j=1}^N$, which allows us to bring various convex optimization tools to solve the problem efficiently.

In addition, we want to induce a sparse network structure from the data and avoid overfitting. Basically, if the coefficients $\boldsymbol{\alpha}_{ji} = \boldsymbol{0}$, then there is no edge (or direct influence) from node $j$ to $i$. For this purpose, we will impose grouped lasso type of regularization on the coefficients $\boldsymbol{\alpha}_{ji}$, i.e., $(\sum_j \|\boldsymbol{\alpha}_{ji}\|)^2$ [16, 19]. Grouped lasso type of regularization has the tendency to select a small number of groups of non-zero coefficients but push other groups of coefficients to be zero. Overall, the optimization problem trades off between the data likelihood term and the group sparsity of the coefficients

$$\min_{\{\boldsymbol{\alpha}_{ji}\}_{j=1}^N} \mathcal{L}_i\left(\{\boldsymbol{\alpha}_{ji}\}_{j=1}^N\right) + \lambda\left(\sum_j \|\boldsymbol{\alpha}_{ji}\|\right)^2, \qquad \text{s.t. } \boldsymbol{\alpha}_{ji} \geq 0, \ \forall j, \qquad (11)$$

where $\lambda$ is the regularization parameter. After we obtain a sparse solution from the above optimization, we obtain partial network structures, each of which centers around a particular node $i$. We can then join all the partial structures together and obtain the overall diffusion network. The corresponding hazard function along each edge can also be obtained from (8).

## 4.3 Optimization

We note that (11) is a nonsmooth optimization problem because of the regularizer. There are many ways to solve the optimization problem, and we will illustrate this using a simple algorithm originating from multiple kernel learning [16, 19]. In this approach, an additional set of variables are introduced to turn the optimization problem into a smooth optimization problem. More specifically, let $\gamma_i \geq 0$ and $\sum_j \gamma_j = 1$. Then using Cauchy-Schwartz inequality, we have $(\sum_j \|\boldsymbol{\alpha}_{ji}\|)^2 = (\sum_j (\|\boldsymbol{\alpha}_{ji}\|/\gamma_j^{1/2})\gamma_j^{1/2})^2 \leq (\sum_j \|\boldsymbol{\alpha}_{ji}\|^2/\gamma_j)(\sum_j \gamma_j) = \sum_j \|\boldsymbol{\alpha}_{ji}\|^2/\gamma_j$, where the equality holds when

$$\gamma_j = \|\boldsymbol{\alpha}_{ji}\| / \sum_j \|\boldsymbol{\alpha}_{ji}\|. \qquad (12)$$

With these additional variables, $\gamma_j$, we can solve an alternative smooth optimization problem, which is jointly convex in both $\boldsymbol{\alpha}_{ji}$ and $\gamma_j$

$$\min_{\{\boldsymbol{\alpha}_{ji}, \gamma_j\}_{j=1}^N} \mathcal{L}_i\left(\{\boldsymbol{\alpha}_{ji}\}_{j=1}^N\right) + \lambda \sum_j \frac{\|\boldsymbol{\alpha}_{ji}\|^2}{\gamma_j}, \ \text{ s.t. } \boldsymbol{\alpha}_{ji} \geq 0, \ \gamma_j \geq 0, \ \sum_j \gamma_j = 1, \ \forall j. \qquad (13)$$

There are many ways to solve the convex optimization problem in (13). In this paper, we used a block coordinate descent approach alternating between the optimization of $\boldsymbol{\alpha}_{ji}$ and $\gamma_j$. More specifically, when we fixed $\boldsymbol{\alpha}_{ji}$, we can obtain the best $\gamma_j$ using the closed form formula in (12); when we fixed $\gamma_j$, we can optimize over $\boldsymbol{\alpha}_{ji}$ using, e.g., a projected gradient method. The overall algorithm pseudocodes are given in Algorithm 1. Moreover, we can speed up the optimization in three ways. First, because the optimization is independent for each node $i$, the overall process can be easily parallelized into $N$ separate sub-problems. Second, we can prune the possible nodes that were never infected before node $i$ in any cascade where $i$ was infected. Third, if we further assume that all the edges from the same node belong to the same type of models, especially when the sample

size is small, the $N$ edges could share a common set of $m$ parameters, and thus we can only estimate $N \times m$ parameters in total.

---

**Algorithm 1:** KernelCascade

---

Initialize the diffusion network $\mathcal{G}$ to be empty;
**for** $i = 1$ *to* $N$ **do**
  Intialize $\{\boldsymbol{\alpha}_{ji}\}_{j=1}^N$ and $\{\gamma_j\}_{j=1}^N$;
  **repeat**
    Update $\{\boldsymbol{\alpha}_{ji}\}_{j=1}^N$ using projected gradient method with $\{\gamma_j\}_{j=1}^N$ from last update;
    Update $\{\gamma_j\}_{j=1}^N$ using formula (12) with $\{\boldsymbol{\alpha}_{ji}\}_{j=1}^N$ from last update;
  **until** *convergence*;
  Extract the sparse neighborhood $\mathcal{N}(i)$ of node $i$ from nonzero $\boldsymbol{\alpha}_{ji}$;
  Join $\mathcal{N}(i)$ to the diffusion network $\mathcal{G}$

---

## 5   Experimental Results

We will evaluate KernelCascade on both realistic synthetic networks and real world networks. We compare it to NETINF [4] and NETRATE [3], and we show that KernelCascade can perform significantly better in terms of both recovering the network structures and the transmission functions.

### 5.1   Synthetic Networks

**Network generation.** We first generate synthetic networks that mimic the structural properties of real networks. These synthetic networks can then be used for simulation of information diffusion. Since the latent networks for generating cascades are known in advance, we can perform detailed comparisons between various methods. We use Kronecker generator [10] to examine two types of networks with directed edges: (i) the core-periphery structure [11], which mimics the information diffusion process in real world networks, and (ii) the Erdős-Rényi random networks.

**Influence function.** For each edge $j \rightarrow i$ in a network $\mathcal{G}$, we will assign it a mixture of two Rayleigh distributions: $f_{ji}(t|\theta, a_1, b_1, a_2, b_2) = \theta R_1(t|a_1, b_1) + (1 - \theta)R_2(t|a_2, b_2)$ where $R_i(t|a_i, b_i) = \frac{2}{t-a_i}\left(\frac{t-a_i}{b_i}\right)^2 \exp\left(-\left(\frac{t-a_i}{b_i}\right)^2\right)$, $t \geqslant a_i$, and $\theta \in (0,1)$ is a mixing proportion. We examine three different parameter settings for the transmission function: (1) all edges in network $\mathcal{G}$ have the same transmission function $p(t) = f(t|0.5, 10, 1, 20, 1)$; (2) all edges in network $\mathcal{G}$ have the same transmission function $q(t) = f(t|0.5, 0, 1, 20, 1)$; and (3) all edges in network $\mathcal{G}$ are uniformly randomly assigned to either $p(t)$ or $q(t)$.

**Cascade generation.** Given a network $\mathcal{G}$ and the collection of transmission functions $f_{ji}$ for each edge, we generate a cascade from $\mathcal{G}$ by randomly choosing a node of $\mathcal{G}$ as the root of the cascade. The root node $j$ is then assigned to time stamp $t_j = 0$. For each neighbor node $i$ pointed by $j$, its event time $t_i$ is sampled from $f_{ji}(t)$. The diffusion process will continue by further infecting the neighbors pointed by node $i$ in a breadth-first fashion until either the overall time exceed the predefined observation time window $T^c$ or there is no new node being infected. If a node is infected more than once by multiple parents, only the first infection time stamp will be recorded.

**Experiment setting and evaluation metric.** We consider a combination of two network topologies (i)-(ii) with three different transmission function settings (1)-(3), which results in six different experimental settings. For each setting, we randomly instantiate the network topologies and transmission functions for 10 times and then vary the number of cascades from $50, 100, 200, 400, 800$ to $1000$. For KernelCascade, we use a Gaussian RBF kernel. The kernel bandwidth $\sigma$ is chosen using median pairwise distance between grid time points. The regularization parameter is chosen using two fold cross-validation. NETINF requires the desired number of edges as input, and we give it an advantage and supply the true number of edges to it. For NETRATE, we experimented with both exponential and Rayleigh transmission function.

We compare different methods in terms of (1) $F1$ score for the network recovery. $F1 := \frac{2 \cdot precision \cdot recall}{precision + recall}$, where $precision$ is the fraction of edges in the inferred network that also present in the true network and $recall$ is the fraction of edges in the true network that also present in the inferred network; (2) $KL$ divergence between the estimated transmission function and the true transmission function, averaged over all edges in a network; (3) the shape of the fitted transmission function compared to the true transmission function.

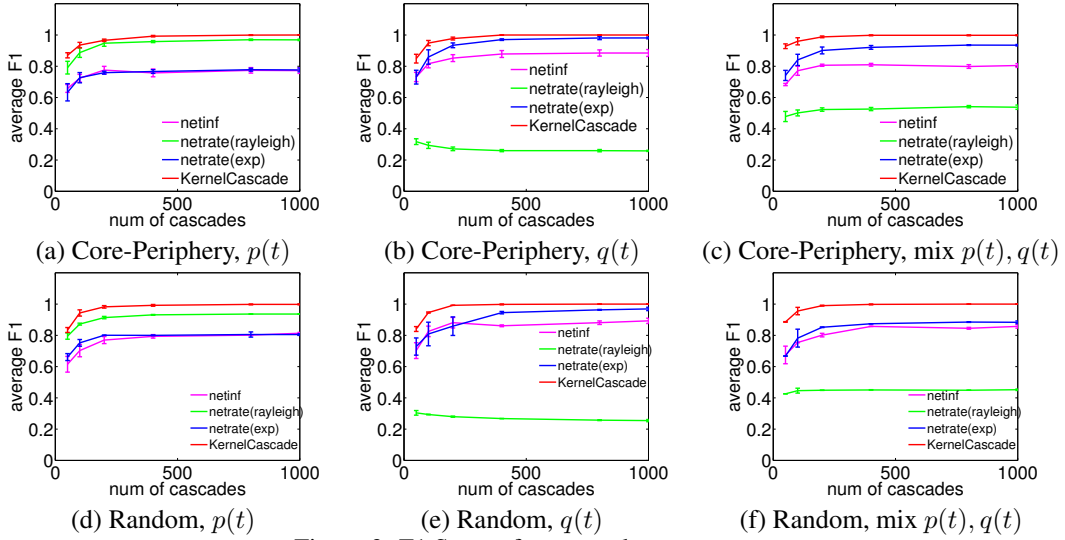

(a) Core-Periphery, $p(t)$     (b) Core-Periphery, $q(t)$     (c) Core-Periphery, mix $p(t), q(t)$

(d) Random, $p(t)$     (e) Random, $q(t)$     (f) Random, mix $p(t), q(t)$

Figure 3: F1 Scores for network recovery.

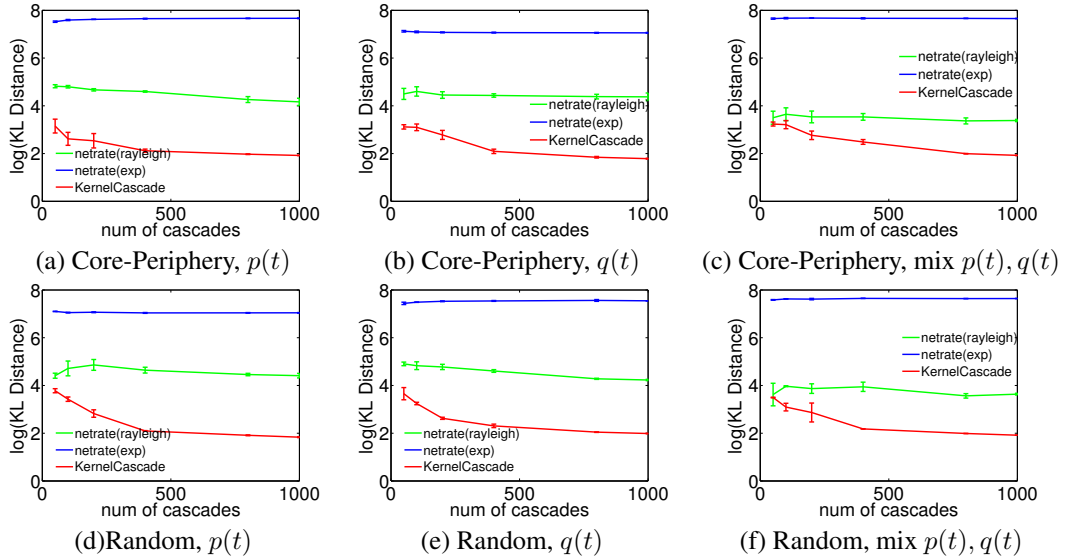

(a) Core-Periphery, $p(t)$     (b) Core-Periphery, $q(t)$     (c) Core-Periphery, mix $p(t), q(t)$

(d)Random, $p(t)$     (e) Random, $q(t)$     (f) Random, mix $p(t), q(t)$

Figure 4: KL Divergence between the estimated and the true transmission function.

**F1 score for network recovery.** From Figure 3, we can see that in all cases, KernelCascade performs consistently and significantly better than NETINF and NETRATE. Furthermore, its performance also steadily increases as we increase the number of cascades, and finally KernelCascade recovers the entire network with around 1000 cascades. In contrast, the competitor methods seldom fully recover the entire network given the same number of cascades. We also note that the performance of NETRATE is very sensitive to the choice of the transmission function (exponential vs. Rayleigh). For instance, depending on the actual data generating process, the performance of NETRATE with Rayleigh model can vary from the second best to the worst.

**KL divergence for transmission function.** Besides better network recovery, KernelCascade also estimates the transmission function better. In all cases we experimented, KernelCascade leads to drastic improvement in recovering the transmission function (Figure 4). We also observe that as we increase the number of cascades, KernelCascade adapts better to the actual transmission function. In contrast, the performance of NETRATE with exponential model does not improve with increasing number of cascades, since the parametric model assumption is incorrect. We note that NETINF does not recover the transmission function, and hence there is no corresponding curve in the plot.

**Visualization of the transmission function.** We also visualize the estimated transmission function for an edge from different methods in Figure 5. We can see that KernelCascade captures the essential

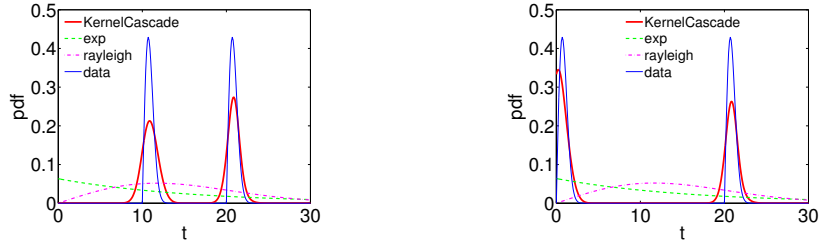

(a) An edge with transmission function $p(t)$     (b) An edge with transmission function $q(t)$

Figure 5: Estimated transmission function of a single edge based on 1000 cascades against the true transmission function (blue curve).

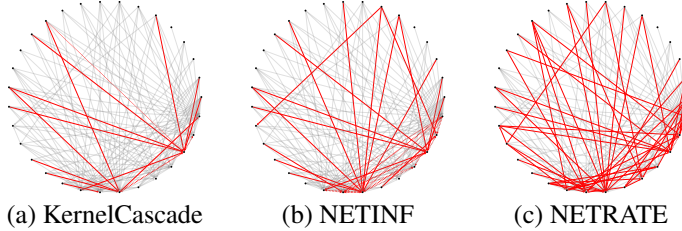

(a) KernelCascade          (b) NETINF          (c) NETRATE

Figure 6: Estimated network of top 32 sites. Edges in grey are correctly uncovered, while edges highlighted in red are either missed or estimated falsely.

features of the true transmission function, *i.e.*, bi-modal behavior, while the competitor methods miss out the important statistical feature completely.

## 5.2    Real world dataset

Finally, we use the MemeTracker dataset [3] to compare NETINF, NETRATE and KernelCascade. In this dataset, the hyperlinks between articles and posts can be used to represent the flow of information from one site to another site. When a site publishes a new post, it will put hyperlinks to related posts in some other sites published earlier as its sources. Later as it also becomes "older", it will be cited by other newer posts as well. As a consequence, all the time-stamped hyperlinks form a cascade for particular piece of information (or event) flowing among different sites. The networks formed by these hyperlinks are used to be the ground truth. We have extracted a network consisting of top 500 sites with 6,466 edges and 11,530 cascades from 7,181,406 posts in a month, and we want to recover the the underlying networks. From Table 1, we can see that KernelCascade achieves a much better $F1$ score for network recovery compared to other methods. Finally, we visualize the estimated sub-network structure for the top 32 sites in Figure 6. By comparison, KernelCascade has a relatively better performance with fewer misses and false predictions.

Table 1: Network recovery results from MemeTracker dataset.

| methods | precision | recall | F1 | predicted edges |
|---|---|---|---|---|
| NETINF | 0.62 | 0.62 | 0.62 | 6466 |
| NETRATE(exp) | 0.93 | 0.23 | 0.37 | 1600 |
| KernelCascade | 0.79 | 0.66 | **0.72** | 5368 |

# 6    Conclusion

In this paper, we developed a flexible kernel method, called KernelCascade, to model the latent diffusion processes and to infer the hidden network with heterogeneous influence between each pair of nodes. In contrast to previous state-of-the-art, such as NETRATE, NETINF and CONNIE, KernelCascade makes no restricted assumption on the specific form of the transmission function over network edges. Instead, it can infer it automatically from the data, which allows each pair of nodes to have a different type of transmission model and better captures the heterogeneous influence among entities. We obtain an efficient algorithm and demonstrate experimentally that KernelCascade can significantly outperforms previous state-of-the-art in both synthetic and real data. In future, we will explore the combination of kernel methods, sparsity inducing norms and other point processes to address a diverse range of social network problems.

**Acknowledgement:** L.S. is supported by NSF IIS-1218749 and startup funds from Gatech.

# References

[1] M. De Choudhury, W. A. Mason, J. M. Hofman, and D. J. Watts. Inferring relevant social networks from interpersonal communication. In *WWW*, pages 301–310, 2010.

[2] N. Eagle, A. S. Pentland, and D. Lazer. From the cover: Inferring friendship network structure by using mobile phone data. *Proceedings of the National Academy of Sciences*, 106(36):15274–15278, Sept. 2009.

[3] M. Gomez-Rodriguez, D. Balduzzi, and B. Schölkopf. Uncovering the temporal dynamics of diffusion networks. In *ICML*, pages 561–568, 2011.

[4] M. Gomez-Rodriguez, J. Leskovec, and A. Krause. Inferring networks of diffusion and influence. In *KDD*, pages 1019–1028, 2010.

[5] D. Kempe, J. M. Kleinberg, and É. Tardos. Maximizing the spread of influence through a social network. In *KDD*, pages 137–146, 2003.

[6] M. Kolar, L. Song, A. Ahmed, and E. Xing. Estimating time-varying networks. *The Annals of Applied Statistics*, 4(1):94–123, 2010.

[7] J. F. Lawless. *Statistical Models and Methods for Lifetime Data*. Wiley-Interscience, 2002.

[8] E. T. Lee and J. Wang. *Statistical Methods for Survival Data Analysis*. Wiley-Interscience, Apr. 2003.

[9] J. Leskovec, L. Backstrom, and J. M. Kleinberg. Meme-tracking and the dynamics of the news cycle. In *KDD*, pages 497–506, 2009.

[10] J. Leskovec, D. Chakrabarti, J. M. Kleinberg, C. Faloutsos, and Z. Ghahramani. Kronecker graphs: An approach to modeling networks. *Journal of Machine Learning Research*, 11:985–1042, 2010.

[11] J. Leskovec, K. J. Lang, and M. W. Mahoney. Empirical comparison of algorithms for network community detection. In *WWW*, pages 631–640, 2010.

[12] J. Leskovec, A. Singh, and J. M. Kleinberg. Patterns of influence in a recommendation network. In *PAKDD*, pages 380–389, 2006.

[13] A. C. Lozano and V. Sindhwani. Block variable selection in multivariate regression and high-dimensional causal inference. In *NIPS*, pages 1486–1494, 2010.

[14] S. A. Myers and J. Leskovec. On the convexity of latent social network inference. In *NIPS*, pages 1741–1749, 2010.

[15] W. Press, S. Teukolsky, W. Vetterling, and B. Flannery. Numerical recipes in C: the art of scientific computing. *Cambridge*, 1992.

[16] A. Rakotomamonjy, F. Bach, S. Canu, Y. Grandvalet, et al. Simplemkl. *Journal of Machine Learning Research*, 9:2491–2521, 2008.

[17] D. Watts and S. Strogatz. Collective dynamics of small-world networks. *Nature*, 393(6684):440–442, June 1998.

[18] D. J. Watts and P. S. Dodds. Influentials, networks, and public opinion formation. *Journal of Consumer Research*, 34(4):441–458, 2007.

[19] Z. Xu, R. Jin, H. Yang, I. King, and M. R. Lyu. Simple and efficient multiple kernel learning by group lasso. In *ICML*, pages 1175–1182, 2010.

